# Bayesian Map Learning in Dynamic Environments

**Kevin P. Murphy**
Computer Science Division
University of California
Berkeley, CA 94720-1776
*murphyk@cs.berkeley.edu*

## Abstract

We consider the problem of learning a grid-based map using a robot with noisy sensors and actuators. We compare two approaches: online EM, where the map is treated as a fixed parameter, and Bayesian inference, where the map is a (matrix-valued) random variable. We show that even on a very simple example, online EM can get stuck in local minima, which causes the robot to get "lost" and the resulting map to be useless. By contrast, the Bayesian approach, by maintaining multiple hypotheses, is much more robust. We then introduce a method for approximating the Bayesian solution, called Rao-Blackwellised particle filtering. We show that this approximation, when coupled with an active learning strategy, is fast but accurate.

## 1 Introduction

The problem of getting mobile robots to autonomously learn maps of their environment has been widely studied (see e.g., [9] for a collection of recent papers). The basic difficulty is that the robot must know exactly where it is (a problem called localization), so that it can update the right part of the map. However, to know where it is, the robot must already have a map: relying on dead-reckoning alone (i.e., integrating the motor commands) is unreliable because of noise in the actuators (slippage and drift).

One obvious solution is to use EM, where we alternate between estimating the location given the map (the E step), and estimating the map given the location (the M step). Indeed, this approach has been successfully used by several groups [8, 11, 12]. However, in all of these works, the trajectory of the robot was specified by hand, and the map was learned off-line. For fully autonomous operation, and to cope with dynamic environments, the map must be learned online.

We consider two approaches to online learning: online EM, and Bayesian inference,

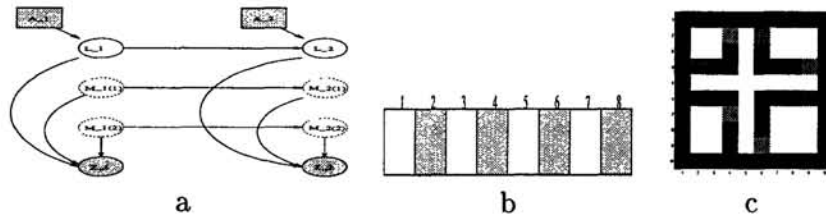

Figure 1: (a) The POMDP represented as a graphical model. $L_t$ is the location, $M_t(i)$ is the label of the $i$'th grid cell, $A_t$ is the action, and $Z_t$ is the observation. Dotted circles denote variables that EM treats as parameters. (b) A one-dimensional grid with binary labels (white = 0, black = 1). (c) A two-dimensional grid, with four labels (closed doors, open doors, walls, and free space).

where we treat the map as a random variable. In Section 3, we show that the Bayesian approach can lead to much better results than online EM; unfortunately, it is computationally intractable, so in Section 4, we discuss an approximation based on Rao-Blackwellised particle filtering.

## 2   The model

We now precisely define the model that we will use in this paper; it is similar to, but much simpler than, the occupancy grid model in [12]. The map is defined to be a grid, where each cell has a label which represents what the robot would see at that point. More formally, the map at time $t$ is a vector of discrete random variables, $M_t(i) \in \{1, \ldots, N_O\}$, where $1 \leq i \leq N_L$. Of course, the map is not observed directly, and nor is the robot's location, $L_t \in \{1, \ldots, N_L\}$. What is observed is $Z_t \in \{1, \ldots, N_O\}$, the label of the cell at the robot's current location, and $A_t \in \{1, \ldots, N_A\}$, the action chosen by the robot just before time $t$. The conditional independence assumptions we are making are illustrated in Figure 1(a). We start by considering the very simple one-dimensional grid shown in Figure 1(b), where there are just two actions, move right ($\rightarrow$) and move left ($\leftarrow$), and just two labels, off (0) and on (1). This is sufficiently small that we can perform exact Bayesian inference. Later, we will generalize to two dimensions.

The prior for the location is a delta function with all its mass on the first (left-most) cell, independent of $A_1$. The transition model for the location is as follows.

$$\Pr(L_t = j | L_{t-1} = i, A_t = \rightarrow) = \begin{cases} p_a & \text{if } j = i+1, j < N \\ 1 - p_a & \text{if } j = i, j < N \\ 1 & \text{if } j = i = N \\ 0 & \text{otherwise} \end{cases}$$

where $p_a$ is the probability of a successful action, i.e., $1 - p_a$ is the probability that the robot's wheels slip. There is an analogous equation for the case when $A_t = \leftarrow$. Note that it is not possible to pass through the "rightmost" cell; the robot can use this information to help localize itself.

The prior for the map is a product of the priors for each cell, which are uniform. (We could model correlation between neighboring cells using a Markov Random Field, although this is computationally expensive.) The transition model for the map is a product of the transition models for each cell, which are defined as follows:

the probability that a 0 becomes a 1 or vice versa is $p_c$ (probability of change), and hence the probability that the cell label remains the same is $1 - p_c$.

Finally, the observation model is

$$\Pr(Z_t = k | M_t = (m_1, \ldots, m_{N_L}), L_t = i) = \begin{cases} p_o & \text{if } m_i = k \\ 1 - p_o & \text{otherwise} \end{cases}$$

where $p_o$ is the probability of a succesful observation, i.e., $1 - p_o$ is the probability of a classification error. Another way of writing this, that will be useful later, is to introduce the dummy deterministic variable, $Z_t'$, which has the following distribution: $\Pr(Z_t' = k | M_t = (m_1, \ldots, m_{N_L}), L_t = i) = \delta(k, m_i)$, where $\delta(a, b) = 1$ if $a = b$ and is 0 otherwise. Thus $Z_t'$ acts just like a multiplexer, selecting out a component of $M_t$ as determined by the "gate" $L_t$. The output of the multiplexer is then passed through a noisy channel, which flips bits with probability $1 - p_o$, to produce $Z_t$.

## 3  Bayesian learning compared to EM

For simplicity, we assume that the parameters $p_o$, $p_a$ and $p_c$, are all known. (In this section, we use $p_o = 0.9$, $p_a = 0.8$ and $p_c = 0$, so the world is somewhat "slippery", but static in appearance.) The state estimation problem is to compute the belief state $\Pr(L_t, M_t | y_{1:t})$, where $Y_t = (Z_t, A_t)$ is the evidence at time $t$; this is equivalent to performing online inference in the graphical model shown in Figure 1(a). Unfortunately, even though we have assumed that the components of $M_t$ are a priori independent, they become correlated by virtue of sharing a common child, $Z_t$. That is, since the true location of the robot is unknown, all of the cells are possible causes of the observation, and they "compete" to "explain" the data. Hence all of the hidden variables become coupled, and the belief state has size $O(N_L 2^{N_L})$.

If the world is static (i.e., $p_c = 0$), we can treat $M$ as a fixed, but unknown, parameter; this can then be combined with the noisy sensor model to define an HMM with the following observation matrix:

$$B(i, k) \stackrel{\text{def}}{=} \Pr(Z_t = k | L_t = i; M) = \sum_j \Pr(Z_t = k | Z_t' = j) \delta(M(i), j)$$

We can then learn $B$ using EM, as in [8, 11, 12]. (We assume for now that the HMM transition matrix is independent of the map, and encodes the known topology of the grid, i.e., the robot can move to any neighboring cell, no matter what its label is. We will lift this restriction in the 2D example.

We can formulate an online version of EM as follows. We use fixed-lag smoothing with a sliding window of length $W$, and compute the expected sufficient statistics (ESS) for the observation matrix within this window as follows: $\theta_t(i, k) = \sum_{\tau = t - W : Z_\tau = k}^{t} \hat{L}_{\tau|t}(i)$, where $\hat{L}_{\tau|t}(i) = \Pr(L_\tau = i | y_{1:t})$. We can compute $\hat{L}$ using the forwards-backwards algorithm, using $\hat{L}_{t-W-1|t-1}$ as the prior. (The initial condition is $\hat{L} = \pi$, where $\pi$ is the (known) prior for $L_0$.) Thus the cost per time step is $O(2W N_L^2)$. In the M step, we normalize each row of $\theta_t + d \times \theta_{t-1}$, where $0 < d < 1$ is a decay constant, to get the new estimate of $B$. We need to downweight the previous ESS since they were computed using out-of-date parameters; in addition, exponential forgetting allows us to handle dynamic environments. [1] discuss some variations on this algorithm.

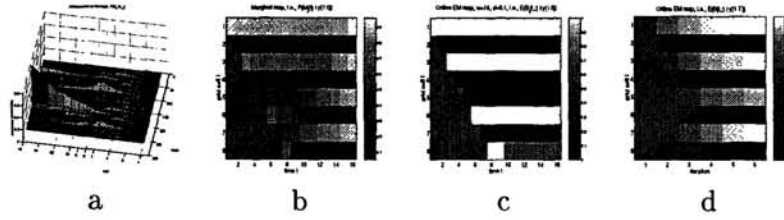

<div align="center">a        b        c        d</div>

Figure 2: (a) The full joint posterior on $P(M_t|y_{1:t})$. 0 and 255, on the axis into the page, represent the maps where every cell is off and every cell is on, respectively; the mode at $t = 16$ is for map 171, which corresponds to the correct pattern 01010101. (b-d) Estimated map. Light cells are more likely to contains 0s, so the correct pattern should have light bars in the odd rows. (b) The marginals of the exact joint. (c) Online EM. (d) Offline EM.

As the window length increases, past locations are allowed to look at more and more future data, and hence their estimates become more accurate; however, the space and time requirements increase. Nevertheless, there are occasions when even the maximum window size (i.e., looking all the way back to $\tau = 0$) will perform poorly, because of the greedy hill-climbing nature of EM. For a simple example of this, consider the environment shown in Figure 1(b). Suppose the robot starts in cell 1, keeps going right until it comes to the end of the "corridor", and then heads back "home". Suppose further that there is a single slippage error at $t = 4$, so the actual path and observation sequence of the robot is as follows:

| $t$ | 1 | 2 | 3 | 4 | 5 | 6 | 7 | 8 | 9 | 10 | 11 | 12 | 13 | 14 | 15 | 16 |
|-----|---|---|---|---|---|---|---|---|---|----|----|----|----|----|----|----|
| $L_t$ | 1 | 2 | 3 | 4 | 4 | 5 | 6 | 7 | 8 | 7 | 6 | 5 | 4 | 3 | 2 | 1 |
| $Z_t$ | 0 | 1 | 0 | 1 | 1 | 0 | 1 | 0 | 1 | 0 | 1 | 0 | 1 | 0 | 1 | 0 |
| $A_t$ | - | → | → | → | → | → | → | → | ← | ← | ← | ← | ← | ← | ← | ← |

To study the effect of this sequence, we computed $\Pr(M_t, L_t|y_{1:t})$ by applying the junction tree algorithm to the graphical model in Figure 1(a). We then marginalized out $L_t$ to compute the posterior $P(M_t)$: see Figure 2(a). At $t = 1$, there are $2^7$ modes, corresponding to all possible bit patterns on the unobserved cells. At each time step, the robot thinks it is moving one step to the right. Hence at $t = 8$, the robot thinks it is in cell 8, and observes 0. When it tries to move right, it knows it will remain in cell 8 (since the robot knows where the boundaries are). Hence at $t = 9$, it is almost 70% confident that it is in cell 8. At $t = 9$, it observes a 1, which contradicts its previous observation of 0. There are two possible explanations: this is a sensor error, or there was a motor error. Which of these is more likely depends on the relative values of the sensor noise, $p_o$, and the system noise, $p_a$. In our experiments, we found that the motor error hypothesis is much more likely; hence the mode of the posterior jumps from the wrong map (in which $M(5) = 1$) to the right map (in which $M(5) = 0$). Furthermore, as the robot returns to "familiar territory", it is able to better localize itself (see Figure 3(a)), and continues to learn the map even for far-away cells, because they are all correlated (in Figure 2(b), the entry for cell 8 becomes sharper even as the robot returns to cell 1)

We now compare the Bayesian solution with EM. Online EM with no smoothing was not able to learn the correct map. Adding smoothing with the maximum window size of $W_t = t$ did not improve matters: it is still unable to escape the local

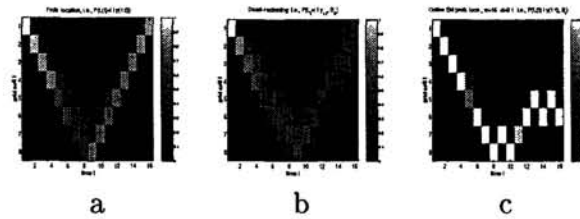

$$\begin{array}{ccc} a & b & c \end{array}$$

Figure 3: Estimated location. Light cells are more likely to contain the robot. (a) Optimal Bayes solution which marginalizes out the map. (b) Dead-reckoning solution which ignores the map. Notice how "blurry" it is. (c) Online EM solution using fixed-lag smoothing with a maximal window length.

minimum in which $M(5) = 1$, as shown in Figure 2(c). (We tried various values of the decay rate $d$, from 0.1 to 0.9, and found that it made little difference.) With the wrong map, the robot "gets lost" on the return journey: see Figure 3(c). Offline EM, on the other hand, does very well, as shown in Figure 2(d); although the initial estimate of location (see Figure 3(b)) is rather diffuse, as it updates the map it can use the benefit of hindsight to figure out where it must have been.

## 4 Rao-Blackwellised particle filtering

Although the Bayesian solution exhibits some desirable properties, its running time is exponential in the size of the environment. In this section, we discuss a sequential Monte Carlo algorithm called particle filtering (also known as SIR filtering, the bootstrap filter, the condensation algorithm, survival of the fittest, etc; see [10, 4] for recent reviews). Particle filtering (PF) has already been successfully applied to the problem of (global) robot localization [5]. However, in that case, the state space was only of dimension 3: the unknowns were the position of the robot, $(x, y) \in \mathbb{R}^2$, and its orientation, $\theta \in [0, 2\pi]$. In our case, the state space is discrete and of dimension $O(1 + N_L)$, since we need to keep track of the map as well as the robot's location (we ignore orientation in this paper).

Particle filtering can be very inefficient in high-dimensional spaces. The key observation which makes it tractable in this context is that, if $L_{1:t}$ were known, then the posterior on $M_t$ would be factored; hence $M_t$ can be marginalized out analytically, and we only need to sample $L_t$. This idea is known in the statistics literature as Rao-Blackwellisation [10, 4]. In more detail, we will approximate the posterior at time $t$ using a set of weighted particles, where each particle specifies a trajectory $L_{1:t}$, and the corresponding conditionally factored representation of $P(M_t) = \prod_i P(M_t(i))$; we will denote the $j$'th particle at time $t$ as $b_t^{(j)}$. Note that we do not need to actually store the complete trajectories $L_{1:t}$: we only need the most recent value of $L$. The approach we take is essentially the same as the one used in the conditional linear Gaussian models of [4, 3], except we replace the Kalman filter update with one which exploits the conditionally factored representation of $P(M_t)$. In particular, the algorithm is as follows: For each particle $j = 1, \ldots, N_s$, we do the following:

1. Sample $L_{t+1}^{(j)}$ from a proposal distribution, which we discuss below.

2. Update each component of the map separately using $L_{t+1}^{(j)}$ and $z_{t+1}$

$$\Pr(M_{t+1}^{(j)} | L_{t+1}^{(j)} = i, b_t^{(j)}, z_{t+1}) \propto \Pr(z_{t+1} | M_{t+1}^{(j)}(i)) \prod_k \Pr(M_{t+1}^{(j)}(k) | M_t^{(j)}(k))$$

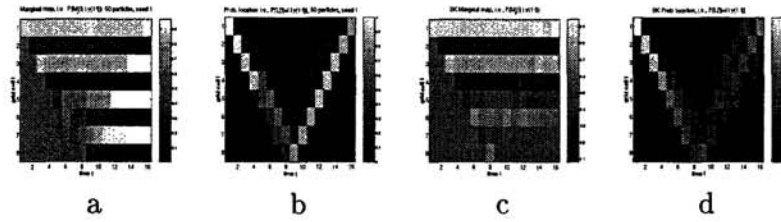

Figure 4: (a-b) Results using 50 particles. (c-d) Results using BK.

3. Update the weights: $w_{t+1}^{(j)} = u_{t+1}^{(j)} w_t^{(j)}$, where $u_{t+1}^{(j)}$ is defined below.

We then resample $N_s$ particles from the normalised weights, using Liu's residual resampling algorithm [10], and set $w_{t+1}^{(j)} = 1/N_s$ for all $j$. We consider two proposal distributions. The first is a simple one which just uses the transition model to predict the new location: $\Pr(L_{t+1}|b_t^{(j)}, a_{t+1})$. In this case, the incremental weight is $u_{t+1}^{(j)} \propto P(z_{t+1}|L_{t+1}^{(j)}, b_t^{(j)})$. The optimal proposal distribution (the one which minimizes the variance of the importance weights) takes the most recent evidence into account, and can be shown to have the form $\Pr(L_{t+1}|b_t^{(j)}, a_{t+1}, z_{t+1})$ with incremental weight $u_{t+1} \propto P(z_{t+1}|b_t^{(j)})$. Computing this requires marginalizing out $M_{t+1}$ and $L_{t+1}$, which can be done in $O(N_L)$ time (details omitted).

In Figure 4, we show the results of applying the above algorithm to the same problem as in Section 3; it can be seen that it approximates the exact solution very closely, using only 50 particles. The results shown are for a particular random number seed; other seeds produce qualitatively very similar results, indicating that 50 particles is in fact sufficient in this case. Obviously, as we increase the number of particles, the error and variance decrease, but the running time increases (linearly).

The question of how many particles to use is a difficult one: it depends both on the noise parameters and the structure of the environment (if every cell has a unique label, localization is easy). Since we are sampling trajectories, the number of hypotheses, and hence the number of particles needed, grows exponentially with time. In the above example, the robot was able to localize itself quite accurately when it reached the end of the corridor, where most hypotheses "died off". In general, the number of particles will depend on the length of the longest cycle in the environment, so we will need to use active learning to ensure tractability.

In the dynamic two-dimensional grid world of Figure 1(c), we chose actions so as to maximize expected discounted reward (using policy iteration), where the reward for visiting cell $i$ is

$$H(L_t)(1 - H(M_t(i))) + (1 - H(L_t))H(M_t(i))$$

where $H(\cdot)$ is the normalized entropy. Hence, if the robot is "lost", so $H(L_t) \approx 1$, the robot will try to visit a cell which it is certain about (see [6] for a better approach); otherwise, it will try to explore uncertain cells. After learning the map, the robot spends its time visiting each of the doors, to keep its knowledge of their state (open or closed) up-to-date.

We now briefly consider some alternative approximate inference algorithms. Examining the graphical structure of our model (see Figure 1(a)), we see that it is identical

to a Factorial HMM [7] (ignoring the inputs). Unfortunately, we cannot use their variational approximation, because they assume a conditional Gaussian observation model, whereas ours is almost deterministic. Another popular approximate inference algorithm for dynamic Bayes nets (DBNs) is the "BK algorithm" [2, 1]. This entails projecting the joint posterior at time $t$ onto a product-of-marginals representation

$$P(L_t, M_t(1), \ldots, M_t(N_L)|y_{1:t}) = P(L_t|y_{1:t}) \prod_i P(M_t(i)|y_{1:t})$$

and using this as a factored prior for Bayesian updating at time $t + 1$. Given a factored prior, we can compute a factored posterior in $O(N_L)$ time by conditioning on each $L_{t+1}$, and then averaging. We found that the BK method does very poorly on this problem (see Figure 4), because it ignores correlation between the cells. Of course, it is possible to use pairwise or higher order marginals for tightly coupled sets of variables. Unfortunately, the running time is exponential in the size of the largest marginal, and in our case, all the $M_t(i)$ variables are coupled.

## Acknowledgments

I would like to thank Nando de Freitas for helping me get particle filtering to work, Sebastian Thrun for an interesting discussion at the conference, and Stuart Russell for encouraging me to compare to EM. This work was supported by grant number ONR N00014-97-1-0941.

## References

[1] X. Boyen and D. Koller. Approximate learning of dynamic models. In *NIPS*, 1998.

[2] X. Boyen and D. Koller. Tractable inference for complex stochastic processes. In *UAI*, 1998.

[3] R. Chen and S. Liu. Mixture Kalman filters. *Submitted*, 1999.

[4] A. Doucet, S. Godsill, and C. Andrieu. On sequential Monte Carlo sampling methods for Bayesian filtering. *Statistics and Computing*, 1999.

[5] D. Fox, W. Burgard, F. Dellaert, and S. Thrun. Monte carlo localization: Efficient position estimation for mobile robots. In *AAAI*, 1999.

[6] D. Fox, W. Burgard, and S. Thrun. Active Markov localization for mobile robots. *Robotics and Autonomous Systems*, 1998.

[7] Z. Ghahramani and M. Jordan. Factorial Hidden Markov Models. *Machine Learning*, 29:245–273, 1997.

[8] S. Koenig and R. Simmons. Unsupervised learning of probabilistic models for robot navigation. In *ICRA*, 1996.

[9] D. Kortenkamp, R. Bonasso, and R. Murphy, editors. *Artificial Intelligence and Mobile Robots: case studies of successful robot systems*. MIT Press, 1998.

[10] J. Liu and R. Chen. Sequential monte carlo methods for dynamic systems. *JASA*, 93:1032–1044, 1998.

[11] H. Shatkay and L. P. Kaelbling. Learning topological maps with weak local odometric information. In *IJCAI*, 1997.

[12] S. Thrun, W. Burgard, and D. Fox. A probabilistic approach to concurrent mapping and localization for mobile robots. *Machine Learning*, 31:29–53, 1998.